# The Product Cut

**Xavier Bresson**
Nanyang Technological University
Singapore
xavier.bresson@ntu.edu.sg

**Thomas Laurent**
Loyola Marymount University
Los Angeles
tlaurent@lmu.edu

**Arthur Szlam**
Facebook AI Research
New York
aszlam@fb.com

**James H. von Brecht**
California State University, Long Beach
Long Beach
james.vonbrecht@csulb.edu

## Abstract

We introduce a theoretical and algorithmic framework for multi-way graph partitioning that relies on a multiplicative cut-based objective. We refer to this objective as the Product Cut. We provide a detailed investigation of the mathematical properties of this objective and an effective algorithm for its optimization. The proposed model has strong mathematical underpinnings, and the corresponding algorithm achieves state-of-the-art performance on benchmark data sets.

## 1 Introduction

We propose the following model for multi-way graph partitioning. Let $\mathcal{G} = (V, W)$ denote a weighted graph, with $V$ its vertex set and $W$ its weighted adjacency matrix. We define the *Product Cut* of a partition $\mathcal{P} = (A_1, \ldots, A_R)$ of the vertex set $V$ as

$$\mathbf{Pcut}(\mathcal{P}) = \frac{\prod_{r=1}^{R} \mathcal{Z}(A_r, A_r^c)}{e^{H(\mathcal{P})}}, \qquad H(\mathcal{P}) = -\sum_{r=1}^{R} \theta_r \log \theta_r, \qquad (1)$$

where $\theta_r = |A_r|/|V|$ denotes the relative size of a set. This model provides a distinctive way to incorporate classical notions of a quality partition. The non-linear, non-local function $\mathcal{Z}(A_r, A_r^c)$ of a set measures its intra- and inter-connectivity with respect to the graph. The *entropic balance* $H(\mathcal{P})$ measures deviations of the partition $\mathcal{P}$ from a collection of sets $(A_1, \ldots, A_R)$ with equal size. In this way, the Product Cut optimization parallels the classical Normalized Cut optimization [10, 15, 13] in terms of its underlying notion of cluster, and it arises quite naturally as a multiplicative version of the Normalized Cut.

Nevertheless, the two models strongly diverge beyond the point of this superficial similarity. We provide a detailed analysis to show that (1) settles the compromise between cut and balance in a fundamentally different manner than classical objectives, such as the Normalized Cut or the Cheeger Cut. The sharp inequalities

$$0 \leq \mathbf{Ncut}(\mathcal{P}) \leq 1 \qquad e^{-H(\mathcal{P})} \leq \mathbf{Pcut}(\mathcal{P}) \leq 1 \qquad (2)$$

succinctly capture this distinction; the Product Cut exhibits a non-vanishing lower bound while the Normalized Cut does not. We show analytically and experimentally that this distinction leads to superior stability properties and performance. From an algorithmic point-of-view, we show how to cast the minimization of (1) as a convex maximization program. This leads to a simple, exact continuous relaxation of the discrete problem that has a clear mathematical structure. We leverage this formulation to develop a monotonic algorithm for optimizing (1) via a sequence of linear programs, and we introduce a randomized version of this strategy that leads to a simple yet highly effective

algorithm. We also introduce a simple version of Algebraic Multigrid (AMG) tailored to our problem that allows us to perform each step of the algorithm at very low cost. On graphs that contain reasonably well-balanced clusters of medium scale, the algorithm provides a strong combination of accuracy and efficiency. We conclude with an experimental evaluation and comparison of the algorithm on real world data sets to validate these claims.

## 2 The Product Cut Model

We begin by introducing our notation and by describing the rationale underlying our model. We use $\mathcal{G} = (V, W)$ to denote a graph on $n$ vertices $V = \{v_1, \ldots, v_n\}$ with weighted edges $W = \{w_{ij}\}_{i,j=1}^n$ that encode similarity between vertices. We denote partitions of the vertex set into $R$ subsets as $\mathcal{P} = (A_1, \ldots, A_R)$, with the understanding that the $A_r \subset V$ satisfy the covering $A_1 \cup \ldots \cup A_R = V$ constraint, the non-overlapping $A_r \cap A_s = \emptyset$, $(r \neq s)$ constraint and the non-triviality $A_r \neq \emptyset$ constraint. We use $f, g, h, u, v$ to denote vertex functions $f : V \to \mathbb{R}$, which we view as functions $f(v_i)$ and $n$-vectors $f \in \mathbb{R}^n$ interchangeably. For a $A \subset V$ we use $|A|$ for its cardinality and $\mathbf{1}_A$ for its indicator function. Finally, for a given graph $\mathcal{G} = (V, W)$ we use $D := \mathrm{diag}(W\mathbf{1}_V)$ to denote the diagonal matrix of weighted vertex degrees.

The starting point for our model arises from a well-known and widely used property of the random walk on a graph. Namely, a random walker initially located in a cluster $A$ is unlikely to leave that cluster quickly [8]. Different approaches of quantifying this intuition then lead to a variety of multi-way partitioning strategies for graphs [11, 12, 1]. The personalized page-rank methodology provides an example of this approach. Following [1], given a scalar $0 < \alpha < 1$ and a non-empty vertex subset $A$ we define

$$\mathbf{pr}_A := M_\alpha^{-1} \mathbf{1}_A / |A| \qquad M_\alpha := \left( \mathrm{Id} - \alpha W D^{-1} \right) / (1 - \alpha) \tag{3}$$

as its personalized page-rank vector. As $\mathbf{1}_A / |A|$ is the uniform distribution on the set $A$ and $WD^{-1}$ is the transition matrix of the random walk on the graph, $\mathbf{pr}_A$ corresponds to the stationary distribution of a random walker that, at each step, moves with probability $\alpha$ to a neighboring vertex by a usual random walk, and has a probability $(1 - \alpha)$ to teleport to the set $A$. If $A$ has a reasonable cluster structure, then $\mathbf{pr}_A$ will concentrate on $A$ and assign low probabilities to its complement. Given a high-quality partition $\mathcal{P} = (A_1, \ldots, A_R)$ of $V$, we therefore expect that $\sigma_{i,r} := \mathbf{pr}_{A_r}(v_i)$ should achieve its maximal value over $1 \leq r \leq R$ when $r = r(i)$ is the class of the $i^{\mathrm{th}}$ vertex.

Viewed from this perspective, we can formulate an $R$-way graph partitioning problem as the task of selecting $\mathcal{P} = (A_1, \ldots, A_R)$ to maximize some combination of the collection $\{\sigma_{i,r(i)} : i \in V\}$ of page-rank probabilities generated by the partition. Two intuitive options immediately come to mind, the arithmetic and geometric means of the collection:

Maximize $\quad \frac{1}{n} \sum_r \sum_{v_i \in A_r} \mathbf{pr}_{A_r}(v_i) \qquad$ over all partitions $(A_1, \ldots, A_R)$ of $V$ into $R$ sets. (4)

Maximize $\quad \left( \prod_r \prod_{v_i \in A_r} \mathbf{pr}_{A_r}(v_i) \right)^{1/n} \quad$ over all partitions $(A_1, \ldots, A_R)$ of $V$ into $R$ sets. (5)

The first option corresponds to a straightforward variant of the classical Normalized Cut. The second option leads to a different type of cut-based objective that we term the Product Cut. The underlying reason for considering (5) is quite natural. If we view each $\mathbf{pr}_{A_r}$ as a probability distribution, then (5) corresponds to a formal likelihood of the partition. This proves quite analogous to re-formulating the classical $k$-means objective for partitioning $n$ data points $(\mathbf{x}_1, \ldots, \mathbf{x}_n)$ into $R$ clusters $(A_1, \ldots, A_R)$ in terms of maximizing a likelihood

$$\prod_{r=1}^R \prod_{v_i \in A_r} \exp\left(-\frac{\|\mathbf{x}_i - \mathbf{m}_r\|^2}{2\sigma_r^2}\right)$$

of Gaussian densities. While the Normalized Cut variant (4) is certainly popular, we show that it suffers from several defects that the Product Cut resolves. As the Product Cut can be effectively optimized and generally leads to higher quality partitions, it therefore provides a natural alternative.

To make these ideas precise, let us define the $\alpha$-smoothed similarity matrix as $\Omega_\alpha := M_\alpha^{-1}$ and use $\{\omega_{ij}\}_{i,j=1}^n$ to denote its entries. Thus $\omega_{ij} = (M_\alpha^{-1} \mathbf{1}_{v_j})_i = \mathbf{pr}_{\{v_j\}}(v_i)$, and so $\omega_{ij}$ gives a non-local measure of similarity between the vertices $v_i$ and $v_j$ by means of the personalized page-rank diffusion process. The matrix $\Omega_\alpha$ is column stochastic, non-symmetric, non-sparse, and has diagonal entries

greater than $(1 - \alpha)$. Given a partition $\mathcal{P} = (A_1, \ldots, A_R)$, we define

$$\textbf{Pcut}(\mathcal{P}) := \frac{\prod_{r=1}^{R} \mathcal{Z}(A_r, A_r^c)^{1/n}}{\mathrm{e}^{H(\mathcal{P})}} \quad \text{and} \quad \textbf{Ncut}(\mathcal{P}) := \frac{1}{R} \sum_{r=1}^{R} \frac{\mathrm{Cut}(A_r, A_r^c)}{\mathrm{Vol}(A_r)} \qquad (6)$$

as its Product Cut and Normalized Cut, respectively. The non-linear, non-local function

$$\mathcal{Z}(A, A^c) := \prod_{v_i \in A_r} 1 + \frac{\sum_{j \in A^c} \omega_{ij}}{\sum_{j \in A} \omega_{ij}} \qquad (7)$$

of a set measures its intra- and inter-connectivity with respect to the graph while $H(\mathcal{P})$ denotes the entropic balance (1). The definitions of

$$\mathrm{Cut}(A, A^c) = \sum_{i \in A_r^c} \sum_{j \in A_r} \omega_{ij} \qquad \text{and} \qquad \mathrm{Vol}(A) = \sum_{i \in V} \sum_{j \in A_r} \omega_{ij}$$

are standard. A simple computation then shows that maximizing the geometric average (5) is equivalent to minimizing the Product Cut, while maximizing the arithmetic average (4) is equivalent to minimizing the Normalized Cut. At a superficial level, both models wish to achieve the same goal. The numerator of the Product Cut aims at a partition in which each vertex is weakly connected to vertices from other clusters and strongly connected with vertices from its own cluster. The denominator $H(\mathcal{P})$ is maximal when $|A_1| = |A_2| = \ldots = |A_R|$, and so aims at a well-balanced partition of the vertices. The objective (5) therefore promotes partitions with strongly intra-connected clusters and weakly inter-connected clusters that have comparable size. The Normalized Cut, defined here on $\Omega_\alpha$ but usually posed over the original similarity matrix $W$, is exceedingly well-known [10, 15] and also aims at finding a good balance between low cut value and clusters of comparable sizes.

Despite this apparent parallel between the Product and Normalized Cuts, the two objectives behave quite differently both in theory and in practice. To illustrate this discrepancy at a high level, note first that the following sharp bounds

$$0 \leq \textbf{Ncut}(\mathcal{P}) \leq 1 \qquad (8)$$

hold for the Normalized Cut. The lower bound is attained for partitions $\mathcal{P}$ in which the clusters are mutually disconnected. For the Product Cut, we have

**Theorem 1** *The following inequality holds for any partition $\mathcal{P}$:*

$$\mathrm{e}^{-H(\mathcal{P})} \leq \textbf{Pcut}(\mathcal{P}) \leq 1. \qquad (9)$$

*Moreover the lower bound is attained for partitions $\mathcal{P}$ in which the clusters are mutually disconnected.*

The lower bound in (9) can be directly read from (6) and (7), while the upper bound is non-trivial and proved in the supplementary material. This theorem goes at the heart of the difference between the Product and Normalized Cuts. To illustrate this, let $\mathcal{P}^{(k)}$ denote a sequence of partitions. Then (9) shows that

$$\lim_{k \to \infty} H(\mathcal{P}^{(k)}) = 0 \Rightarrow \lim_{k \to \infty} \textbf{Pcut}(\mathcal{P}^{(k)}) = 1. \qquad (10)$$

In other words, an arbitrarily ill-balanced partition leads to arbitrarily poor values of its Product Cut. The Normalized Cut does not possess this property. As an extreme but easy-to-analyze example, consider the case where $\mathcal{G} = (V, W)$ is a collection of isolated vertices. All possible partitions $\mathcal{P}$ consist of mutually disconnected clusters and the lower bound is reached for both (8) and (9). Thus $\textbf{Ncut}(\mathcal{P}) = 0$ for all $\mathcal{P}$ and so all partitions are equivalent for the Normalized Cut. On the other hand $\textbf{Pcut}(\mathcal{P}) = \mathrm{e}^{-H(\mathcal{P})}$, which shows that, in the absence of "cut information," the Product Cut will choose the partition that maximizes the entropic balance. So in this case, any partition $\mathcal{P}$ for which $|A_1| = \ldots = |A_R|$ will be a minimizer. In essence, this tighter lower bound for the Product Cut reflects its stronger balancing effect vis-a-vis the Normalized Cut.

## 2.1 (In-)Stability Properies of the Product Cut and Normalized Cut

In practice, the stronger balancing effect of the Product Cut manifests as a stronger tolerance to perturbations. We now delve deeper and contrast the two objectives by analyzing their stability properties using experimental data as well as a simplified model problem that isolates the source of

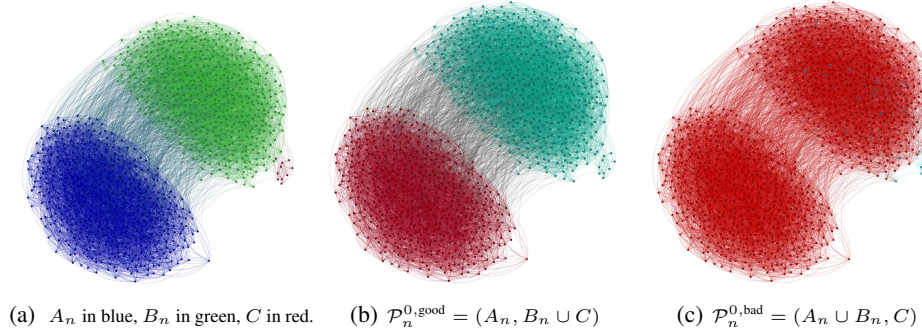

(a) $A_n$ in blue, $B_n$ in green, $C$ in red.     (b) $\mathcal{P}_n^{0,\text{good}} = (A_n, B_n \cup C)$     (c) $\mathcal{P}_n^{0,\text{bad}} = (A_n \cup B_n, C)$

Figure 1: The graphs $\mathcal{G}_n^0$ used for analyzing stability.

the inherent difficulties. Invoking ideas from dynamical systems theory, we say an objective is *stable* if an infinitesimal perturbation of a graph $\mathcal{G} = (V, W)$ leads to an infinitesimal perturbation of the optimal partition. If an infinitesimal perturbation leads to a dramatic change in the optimal partition, then the objective is *unstable*.

We use a simplified model to study stability of the Product Cut and Normalized Cut objectives. Consider a graph $\mathcal{G}_n = (V_n, W_n)$ made of two clusters $A_n$ and $B_n$ containing $n$ vertices each. Each vertex in $\mathcal{G}_n$ has degree $k$ and is connected to $\mu k$ vertices in the opposite cluster, where $0 \le \mu \le 1$. The graph $\mathcal{G}_n^0$ is a perturbation of $\mathcal{G}_n$ constructed by adding a small cluster $C$ of size $n_0 \ll n$ to the original graph. Each vertex of $C$ has degree $k_0$ and is connected to $\mu_0 k_0$ vertices in $B_n$ and $(1 - \mu_0)k_0$ vertices in $C$ for some $0 \le \mu_0 \le 1$. In the perturbed graph $\mathcal{G}_n^0$, a total of $n_0$ vertices in $B_n$ are linked to $C$ and have degree $k + \mu_0 k_0$. See figure 1(a). The main properties of $\mathcal{G}_n, \mathcal{G}_n^0$ are

- Unperturbed graph $\mathcal{G}_n$ :    $|A_n| = |B_n| = n$,    $\text{Cond}_{\mathcal{G}_n}(A_n) = \mu$,   $\text{Cond}_{\mathcal{G}_n}(B_n) = \mu$
- Perturbed graph $\mathcal{G}_n^0$:      $|A_n| = |B_n| = n$,    $\text{Cond}_{\mathcal{G}_n^0}(A_n) = \mu$,   $\text{Cond}_{\mathcal{G}_n^0}(B_n) \approx \mu$
  $|C| = n_0 \ll n$,       $\text{Cond}_{\mathcal{G}_n^0}(C) = \mu_0$.

where $\text{Cond}_{\mathcal{G}}(A) = \text{Cut}(A, A^c)/\min(|A|, |A^c|)$ denotes the conductance of a set. If we consider the parameters $\mu, \mu_0, k, k_0, n_0$ as fixed and look at the perturbed graph $\mathcal{G}_n^0$ in the limit $n \to \infty$ of a large number of vertices, then as $n$ becomes larger the degree of the bulk vertices will remain constant while the size $|C|$ of the perturbation becomes infinitesimal.

To examine the influence of this infinitesimal perturbation for each model, let $\mathcal{P}_n = (A_n, B_n)$ denote the desired partition of the unperturbed graph $\mathcal{G}_n$ and let $\mathcal{P}_n^{0,\text{good}} = (A_n, B_n \cup C)$ and $\mathcal{P}_n^{0,\text{bad}} = (A_n \cup B_n, C)$ denote the partitions of the perturbed graph $\mathcal{G}_n^0$ depicted in figure 1(b) and 1(c), respectively. As $\mathcal{P}_n^{0,\text{good}} \approx \mathcal{P}_n$, a stable objective will prefer $\mathcal{P}_n^{0,\text{good}}$ to $\mathcal{P}_n^{0,\text{bad}}$ while any objective preferring the converse is unstable. A detailed study of stability proves possible for this specific graph family. We summarize the conclusions of this analysis in the theorem below, which shows that the Normalized Cut is unstable in certain parameter regimes while the Product Cut is always stable. The supplementary material contains the proof.

**Theorem 2** *Suppose that $\mu, \mu_0, k, k_0, n_0$ are fixed. Then*

$$\mu_0 < 2\mu \quad \Rightarrow \quad \mathbf{Ncut}_{\mathcal{G}_n^0}(\mathcal{P}_n^{0,good}) > \mathbf{Ncut}_{\mathcal{G}_n^0}(\mathcal{P}_n^{0,bad}) \quad \textit{for } n \textit{ large enough.} \tag{11}$$

$$\mathbf{Pcut}_{\mathcal{G}_n^0}(\mathcal{P}_n^{0,good}) < \mathbf{Pcut}_{\mathcal{G}_n^0}(\mathcal{P}_n^{0,bad}) \quad \textit{for } n \textit{ large enough.} \tag{12}$$

Statement (11) simply says that the large cluster $A_n$ must have a conductance $\mu$ at least twice better than the conductance $\mu_0$ of the small perturbation cluster $C$ in order to prevent instability. Thus adding an infinitesimally small cluster with mediocre conductance (up to two times worse the conductance of the main structure) has the potential of radically changing the partition selected by the Normalized Cut. Moreover, this result holds for the classical Normalized Cut, its smoothed variant (4) as well as for similar objectives such as the Cheeger Cut and Ratio Cut. Conversely, (12) shows that adding an infinitesimally small cluster will not affect the partition selected by the

| | Partition $\mathcal{P}$ of WEBKB4 found by the **Pcut** algo. | Partition $\mathcal{P}$ of WEBKB4 found by the **Ncut** algo. | Partition $\mathcal{P}$ of CITESEER found by the **Pcut** algo. | Partition $\mathcal{P}$ of CITESEER found by the **Ncut** algo. |
|---|---|---|---|---|
| $\mathrm{e}^{-H(\mathcal{P})}$ | .2506 | .7946 | .1722 | .7494 |
| **Pcut**$(\mathcal{P})$ | **.5335** | .8697 | **.4312** | .8309 |
| **Ncut**$(\mathcal{P})$ | .5257 | **.5004** | .5972 | **.5217** |

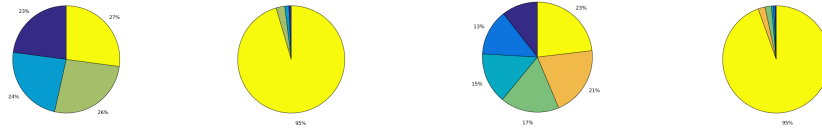

Figure 2: The Product and Normalized Cuts on WEBKB4 ($R = 4$ clusters) and CITESEER ($R = 6$ clusters). The pie charts visually depict the sizes of the clusters in each partition. In both cases, NCut returns a super-cluster while PCut returns a well-balanced partition. The NCut objective prefers the ill-balanced partitions while the PCut objective dramatically prefers the balanced partitions.

Product Cut. The proof, while lengthy, is essentially just theorem 1 in disguise. To see this, note that the sequence of partitions $\mathcal{P}_n^{0,\mathrm{bad}}$ becomes arbitrarily ill-balanced, which from (10) implies $\lim_{n\to\infty} \mathbf{Pcut}_{\mathcal{G}_n^0}(\mathcal{P}_n^{0,\mathrm{bad}}) = 1$. However, the unperturbed graph $\mathcal{G}_n$ grows in a self-similar fashion as $n \to \infty$ and so the Product Cut of $\mathcal{P}_n$ remains approximately a constant, say $\gamma$, for all $n$. Thus $\mathbf{Pcut}_{\mathcal{G}_n}(\mathcal{P}_n) \approx \gamma < 1$ for $n$ large enough, and $\mathbf{Pcut}_{\mathcal{G}_n^0}(\mathcal{P}_n^{0,\mathrm{good}}) \approx \mathbf{Pcut}_{\mathcal{G}_n}(\mathcal{P}_n)$ since $|C|$ is infinitesimal. Therefore $\mathbf{Pcut}_{\mathcal{G}_n^0}(\mathcal{P}_n^{0,\mathrm{good}}) \approx \gamma < 1$. Comparing this upper-bound with the fact $\lim_{n\to\infty} \mathbf{Pcut}_{\mathcal{G}_n^0}(\mathcal{P}_n^{0,\mathrm{bad}}) = 1$, we see that the Product Cut of $\mathcal{P}_n^{0,\mathrm{bad}}$ becomes eventually larger than the Product Cut of $\mathcal{P}_n^{0,\mathrm{good}}$. While we execute this program in full only for the example above, this line of argument is fairly general and similar stability estimates are possible for more general families of graphs.

This general contrast between the Product Cut and the Normalized Cut extends beyond the realm of model problems, as the user familiar with off-the-shelf NCut codes likely knows. When provided with "dirty" graphs, for example an e-mail network or a text data set, NCut has the aggravating tendency to return a super-cluster. That is, NCut often returns a partition $\mathcal{P} = (A_1, \ldots, A_R)$ where a single set $|A_r|$ contains the vast majority of the vertices. Figure 2 illustrates this phenomenon. It compares the partitions obtained for NCut (computed on $\Omega_\alpha$ using a modification of the standard spectral approximation from [15]) and for PCut (computed using the algorithm presented in the next section) on two graphs constructed from text data sets. The NCut algorithm returns highly ill-balanced partitions containing a super-cluser, while PCut returns an accurate and well-balanced partition. Other strategies for optimizing NCut obtain similarly unbalanced partitions. As an example, using the algorithm from [9] with the original sparse weight matrix $W$ leads to relative cluster sizes of $99.2\%$, $0.5\%$, $0.2\%$ and $0.1\%$ for WEBKB4 and $98.5\%$, $0.4\%$, $0.3\%$, $0.3\%$, $0.3\%$ and $0.2\%$ for CITESEER. As our theoretical results indicate, these unbalanced partitions result from the normalized cut criterion itself and not the algorithm used to minimize it.

## 3 The Algorithm

Our strategy for optimizing the Product Cut relies on a popular paradigm for discrete optimization, i.e. *exact relaxation*. We begin by showing that the discrete, graph-based formulation (5) can be relaxed to a continuous optimization problem, specifically a convex maximization program. We then prove that this relaxation is *exact*, in the sense that optimal solutions of the discrete and continuous problems coincide. With an exact relaxation in hand, we may then appeal to continuous optimization strategies (rather than discrete or greedy ones) for optimizing the Product Cut. This general idea of exact relaxation is intimately coupled with convex maximization.

Assume that the graph $\mathcal{G} = (V, W)$ is connected. Then by taking the logarithm of (5) we see that (5) is equivalent to the problem

$$\left. \begin{array}{l} \text{Maximize} \quad \sum_{r=1}^{R} \sum_{i \in A_r} \log \frac{(\Omega_\alpha \mathbf{1}_{A_r})_i}{|A_r|} \\ \text{over all partitions } \mathcal{P} = (A_1, \ldots, A_R) \text{ of } V \text{ into } R \text{ non-empty subsets.} \end{array} \right\} \quad \text{(P)}$$

The relaxation of (P) then follows from the usual approach. We first encode sets $A_r \subsetneq V$ as binary vertex functions $\mathbf{1}_{A_r}$, then relax the binary constraint to arrive at a continuous program. Given a vertex function $f \in \mathbb{R}^n_+$ with non-negative entries, we define the continuous energy $e(f)$ as

$$e(f) := \langle f, \log \left( \Omega_\alpha f / \langle f, \mathbf{1}_V \rangle \right) \rangle \quad \text{if } f \neq 0, \quad \text{and} \quad e(0) = 0,$$

where $\langle \cdot, \cdot \rangle$ denotes the usual dot product in $\mathbb{R}^n$ and the logarithm applies entriwise. As $(\Omega_\alpha f)_i > 0$ whenever $f \neq 0$, the continuous energy is well-defined. After noting that $\sum_r e(\mathbf{1}_{A_r})$ is simply the objective value in problem (P), we arrive to the following continuous relaxation

$$\left. \begin{array}{c} \text{Maximize } \sum_{r=1}^R e(f_r) \\ \text{over all } (f_1, \dots, f_R) \in \mathbb{R}^n_+ \times \dots \times \mathbb{R}^n_+ \text{ satisfying } \sum_{r=1}^R f_r = \mathbf{1}_V \end{array} \right\}, \quad \text{(P-rlx)}$$

where the non-negative cone $\mathbb{R}^n_+$ consists of all vectors in $\mathbb{R}^n$ with non-negative entries.

The following theorem provides the theoretical underpinning for our algorithmic approach. It establishes convexity of the relaxed objective for connected graphs.

**Theorem 3** *Assume that $\mathcal{G} = (V, W)$ is connected. Then the energy $e(f)$ is continuous, positive 1-homogeneous and convex on $\mathbb{R}^n_+$. Moreover, the strict convexity property*

$$e(\theta f + (1 - \theta)g) < \theta e(f) + (1 - \theta)e(g) \quad \text{for all} \quad \theta \in (0, 1)$$

*holds whenever $f, g \in \mathbb{R}^n_+$ are linearly independent.*

The continuity of $e(f)$ away from the origin as well as the positive one-homogeneity are obvious, while the continuity of $e(f)$ at the origin is easy to prove. The proof of convexity of $e(f)$, provided in the supplementary material, is non-trivial and heavily relies on the particular structure of $\Omega_\alpha$ itself. With convexity of $e(f)$ in hand, we may prove the main theorem of this section.

**Theorem 4 (** Equivalence of (P) and (P-rlx) **)** *Assume that $\mathcal{G} = (V, W)$ is connected and that $V$ contains at least $R$ vertices. If $\mathcal{P} = (A_1, \dots, A_R)$ is a global optimum of (P) then $(\mathbf{1}_{A_1}, \dots, \mathbf{1}_{A_R})$ is a global optimum of (P-rlx). Conversely, if $(f_1, \dots, f_R)$ is a global optimum of (P-rlx) then $(f_1, \dots, f_R) = (\mathbf{1}_{A_1}, \dots, \mathbf{1}_{A_R})$ where $(A_1, \dots, A_R)$ is a global optimum of (P).*

**Proof.** By strict convexity, the solution of the maximization (P-rlx) occurs at the extreme points of the constraint set $\Sigma = \{(f_1, \dots, f_R) : f_r \in \mathbb{R}^N_+ \text{ and } \sum_{r=1}^R f_r = \mathbf{1}\}$. Any such extreme point takes the form $(\mathbf{1}_{A_1}, \dots, \mathbf{1}_{A_R})$, where necessarily $A_1 \cup \dots \cup A_R = V$ and $A_r \cap A_s = \emptyset$ $(r \neq s)$ hold. It therefore suffices to rule out extreme points that have an empty set of vertices. But if $A \neq B$ are non-empty then $\mathbf{1}_A, \mathbf{1}_B$ are linearly independent, and so the inequality $e(\mathbf{1}_A + \mathbf{1}_B) < e(\mathbf{1}_A) + e(\mathbf{1}_B)$ holds by strict convexity and one-homogeneity. Thus given a partition of the vertices into $R - 1$ non-empty subsets and one empty subset, we can obtain a better energy by splitting one of the non-empty vertex subsets into two non-empty subsets. Thus any globally maximal partition cannot contain empty subsets. $\square$

With theorems 3 and 4 in hand, we may now proceed to optimize (P) by searching for optima of its exact relaxation. We tackle the latter problem by leveraging **sequential linear programming** or **gradient thresholding** strategies for convex maximization. We may write (P-rlx) as

$$\text{Maximize } \mathcal{E}(F) \quad \text{subject to} \quad F \in C \text{ and } \psi_i(F) = 0 \text{ for } i = 1, \dots, n \quad (13)$$

where $F = (f_1, \dots, f_R)$ is the optimization variable, $\mathcal{E}(F)$ is the convex energy to be maximized, $C$ is the bounded convex set $[0, 1]^n \times \dots \times [0, 1]^n$ and the $n$ affine constraints $\psi_i(F) = 0$ correspond to the row-stochastic constraints $\sum_{r=1}^R f_{i,r} = 1$. Given a current feasible estimate $F^k$ of the solution, we obtain the next estimate $F^{k+1}$ by solving the linear program

$$\text{Maximize } L_k(F) \quad \text{subject to} \quad F \in C \text{ and } \psi_i(F) = 0 \text{ for } i = 1, \dots, n \quad (14)$$

where $L_k(F) = \mathcal{E}(F^k) + \langle \nabla \mathcal{E}(F^{(k)}), F - F^k \rangle$ is the linearization of the energy $\mathcal{E}(F)$ around the current iterate. By convexity of $\mathcal{E}(F)$, this strategy monotonically increases $\mathcal{E}(F^k)$ since $\mathcal{E}(F^{k+1}) \geq L_k(F^{k+1}) \geq L_k(F^k) = \mathcal{E}(F^k)$. The iterates $F^k$ therefore encode a sequence of partitions of $V$ that monotonically increase the energy at each step. Either the current iterate maximizes the linear form, in which case first-order optimality holds, or else the subsequent iterate produces a partition with a

---

**Algorithm 1** Randomized SLP for PCut

---

**Initialization:** $(f_1^0, \ldots, f_R^0) = (\mathbf{1}_{A_1}, \ldots, \mathbf{1}_{A_R})$ for $(A_1, \ldots, A_R)$ a random partition of $V$

**for** $k = 0$ to maxiter **do**

    **for** $r = 1$ to $R$ **do**

        Set $\hat{f}_r = f_r^k / (\sum_{i=1}^n f_{i,r}^k)$           then solve $M_\alpha u_r = \hat{f}_r$

        Set $g_{i,r} = f_{i,r}/u_{i,r}$ for $i = 1, \ldots n$   then solve $M_\alpha^T v_r = g_r$

        Set $h_r = \log u_r + v_r - \mathbf{1}$

    **end for**

    Choose at random $s_k$ vertices and let $\mathcal{I} \subset V$ be these vertices.

    **for** all $i \in V$ **do**

        If $i \in \mathcal{I}$ then $f_{i,r}^{k+1} = \begin{cases} 1 & \text{if } r = \arg\max_s h_{is} \\ 0 & \text{otherwise,} \end{cases}$      if $i \notin \mathcal{I}$ then $f_{i,r}^{k+1} = \begin{cases} 1 & \text{if } h_{i,r} > 0 \\ 0 & \text{otherwise.} \end{cases}$

    **end for**

**end for**

---

strictly larger objective value. The latter case can occur only a finite number of times, as only a finite number of partitions exist. Thus the sequence $F^k$ converges after a finite number of iterations.

While simple and easy to implement, this algorithm suffers from a severe case of early termination. When initialized from a random partition, the iterates $F^k$ almost immediately converge to a poor-quality solution. We may rescue this poor quality algorithm and convert it to a highly effective one, while maintaining its simplicity, by randomizing the LP (14) at each step in the following way. At step $k$ we solve the LP

$$\text{maximize } L_k(F) \quad \text{subject to} \quad F \in C \quad \text{and} \quad \psi_i(F) = 0 \text{ for } i \in \mathcal{I}_k, \tag{15}$$

where the set $\mathcal{I}_k$ is a random subset of $\{1, 2, \ldots, n\}$ obtained by drawing $s_k$ constraints uniformly at random without replacement. The LP (15) is therefore version of LP (14) in which we have dropped a random set of constraints. If we start by enforcing a small number $s_k$ of constraints and slowly increment this number $s_{k+1} = s_k + \Delta s_k$ as the algorithm progresses, we allow the algorithm time to explore the energy landscape. Enforcing more constraints as the iterates progress ensures that (15) eventually coincides with (14), so convergence of the iterates $F^k$ of the randomized algorithm is still guaranteed. The attraction is that LP (15) has a simple, closed-form solution given by a variant of gradient thresholding. We derive the closed form solution of LP (15) in section 1 of the supplementary material, and this leads to Algorithm 1 above.

The overall effectiveness of this strategy relies on two key ingredients. The first is a proper choice of the number of constraints $s_k$ to enforce at each step. Selecting the rate at which $s_k$ increases is similar, in principle, to selecting a learning rate schedule for a stochastic gradient descent algorithm. If $s_k$ increases too quickly then the algorithm will converge to poor-quality partitions. If $s_k$ increases too slowly, the algorithm will find a quality solution but waste computational effort. A good rule of thumb is to linearly increase $s_k$ at some constant rate $\Delta s_k \equiv \lambda$ until all constraints are enforced, at which point we switch to the deterministic algorithm and terminate the process at convergence. The second key ingredient involves approximating solutions to the linear system $M_\alpha x = b$ quickly. We use a simple Algebraic Multigrid (AMG) technique, i.e. a stripped-down version of [7] or [6], to accomplish this. The main insight here is that exact solutions of $M_\alpha x = b$ are not needed, but not all approximate solutions are effective. We need an approximate solution $x$ that has non-zero entries on all of $|V|$ for thresholding to succeed, and this can be accomplished by AMG at very little cost.

## 4 Experiments

We conclude our study of the Product Cut model by presenting extensive experimental evaluation of the algorithm[1]. We intend these experiments to highlight the fact that, in addition to a strong theoretical model, the algorithm itself leads to state-of-the-art performance in terms of cluster purity on a variety of real world data sets. We provide experimental results on four text data sets (20NEWS, RCV1, WEBKB4, CITESEER) and four data sets containing images of handwritten digits (MNIST, PENDIGITS, USPS, OPTDIGITS). We provide the source for these data sets and details on their

Table 1: Algorithmic Comparison via Cluster Purity.

| | 20NE | RCV1 | WEBK | CITE | MNIS | PEND | USPS | OPTI |
|---|---|---|---|---|---|---|---|---|
| size | 20K | 9.6K | 4.2K | 3.3K | 70K | 11K | 9.3K | 5.6K |
| R | 20 | 4 | 4 | 6 | 10 | 10 | 10 | 10 |
| RND | 6 | 30 | 39 | 22 | 11 | 12 | 17 | 12 |
| NCUT | 27 | 38 | 40 | 23 | 77 | 80 | 72 | 91 |
| LSD | 34 | 38 | 46 | 53 | 76 | 86 | 70 | 91 |
| MTV | 36 | 43 | 45 | 43 | 96 | **87** | 85 | 95 |
| GRACLUS | 42 | 42 | 49 | 54 | **97** | 85 | 87 | 94 |
| NMFR | **61** | 43 | **58** | 63 | **97** | **87** | 86 | **98** |
| PCut (.9,$\lambda_1$) | **61** | **53** | **58** | 63 | **97** | **87** | 89 | **98** |
| PCut (.9,$\lambda_2$) | 60 | 50 | 57 | **64** | 96 | 84 | **89** | 95 |

construction in the supplementary material. We compare our method against partitioning algorithms that, like the Product Cut, rely on graph-cut objective principles and that partition the graph in a direct, non-recursive manner. The NCut algorithm [15] is a widely used spectral algorithm that relies on a post-processing of the eigenvectors of the graph Laplacian to optimize the Normalized Cut energy. The NMFR algorithm [14] uses a graph-based random walk variant of the Normalized Cut. The LSD algorithm [2] provides a non-negative matrix factorization algorithm that relies upon a trace-based relaxation of the Normalized Cut objective. The MTV algorithm from [3] and the balanced $k$-cut algorithm from [9] provide total-variation based algorithms that attempt to find an optimal multi-way Cheeger cut of the graph by using $\ell^1$ optimization techniques. Both algorithms optimize the same objective and achieve similar purity values. We report results for [3] only. The GRACLUS algorithm [4, 5] uses a multi-level coarsening approach to optimize the NCut objective as formulated in terms of kernel $k$-means. Table 1 reports the accuracy obtained by these algorithms for each data set. We use cluster purity to quantify the quality of the calculated partition, defined according to the relation: Purity $= \frac{1}{n} \sum_{r=1}^{R} \max_{1<i<R} m_{r,i}$. Here $m_{r,i}$ denotes the number of data points in the $r^{\text{th}}$ cluster that belong to the $i^{\text{th}}$ ground-truth class. The third row of the table (RND) provides a base-line purity for reference, i.e. the purity obtained by assigning each data point to a class from 1 to $R$ uniformly at random. The PCut, MTV and GRACLUS algorithms rely on randomization, so for these algorithms we report the average purity achieved over 500 different runs. For the PCut algorithm, we use $\alpha = .9$ when defining $\Omega_\alpha$. Also, in order to illustrate the tradeoff when selecting the rate at which the number of enforced constraints $s_k$ increases, we report accuracy results for the linear rates

$$\Delta s_k = 10^{-4} \times n := \lambda_1 \qquad \text{and} \qquad \Delta s_k = 5 \times 10^{-4} \times n := \lambda_2$$

where $n$ denotes the total number of vertices in the data set. By and large both PCut and NMFR consistently outperform the other algorithms in terms of accuracy.

Table 2: Computational Time

| MNIST | | | 20NEWS | | |
|---|---|---|---|---|---|
| **NMFR** | **PCut** (.9,$\lambda_1$) | **PCut** (.9,$\lambda_2$) | **NMFR** | **PCut** (.9,$\lambda_1$) | **PCut** (.9,$\lambda_2$) |
| 4.6mn | 11s | 10s | 3.7mn | 1.3mn | 16s |
| (92%) | (92%) | (91%) | (58%) | (58%) | (57%) |

In addition to the accuracy comparisons, table 2 records the time required for PCut and NMFR to reach 95% of their limiting purity value on the two largest data sets, 20NEWS and MNIST. Each algorithm is implemented in a fair and consistent way, and the experiments were all performed on the same architecture. Timing results on the smaller data sets from table 1 are consistent with those obtained for 20NEWS and MNIST. In general we observe that PCut runs significantly faster. Additionally, as we expect for PCut, the slower rate $\lambda_1$ generally leads to more accurate results while the larger rate $\lambda_2$ typically converges more quickly.

When taken together, our theoretical and experimental results clearly reveal that the model provides a promising method for graph partitioning. The algorithm consistently achieves state-of-the-art results, and typically runs significantly faster than other algorithms that achieve a comparable level of accuracy. Additionally, both the model and algorithmic approach rely upon solid mathematical foundations that are frequently missing in the multi-way clustering literature.

**Acknowledgements:** TL was supported by NSF DMS-1414396.

## Footnotes

[1]The code is available at https://github.com/xbresson/pcut

# References

[1] Reid Andersen, Fan Chung, and Kevin Lang. Local graph partitioning using pagerank vectors. In *Proceedings of the 47th Annual Symposium on Foundations of Computer Science (FOCS '06)*, pages 475–486, 2006.

[2] Raman Arora, M Gupta, Amol Kapila, and Maryam Fazel. Clustering by left-stochastic matrix factorization. In *International Conference on Machine Learning (ICML)*, pages 761–768, 2011.

[3] Xavier Bresson, Thomas Laurent, David Uminsky, and James von Brecht. Multiclass total variation clustering. In *Advances in Neural Information Processing Systems (NIPS)*, 2013.

[4] Inderjit S. Dhillon, Yuqiang Guan, and Brian Kulis. Weighted graph cuts without eigenvectors: A multilevel approach. *IEEE Transactions on Pattern Analysis and Machine Intelligence*, 29(11):1944–1957, 2007.

[5] George Karypis and Vipin Kumar. A fast and high quality multilevel scheme for partitioning irregular graphs. *SIAM J. Sci. Comput.*, 20(1):359–392, 1998.

[6] Dilip Krishnan, Raanan Fattal, and Richard Szeliski. Efficient preconditioning of laplacian matrices for computer graphics. *ACM Transactions on Graphics (TOG)*, 32(4):142, 2013.

[7] Oren E Livne and Achi Brandt. Lean algebraic multigrid (lamg): Fast graph laplacian linear solver. *SIAM Journal on Scientific Computing*, 34(4):B499–B522, 2012.

[8] László Lovász and Miklós Simonovits. Random walks in a convex body and an improved volume algorithm. *Random structures & algorithms*, 4(4):359–412, 1993.

[9] Syama Sundar Rangapuram, Pramod Kaushik Mudrakarta, and Matthias Hein. Tight continuous relaxation of the balanced k-cut problem. In *Advances in Neural Information Processing Systems*, pages 3131–3139, 2014.

[10] J. Shi and J. Malik. Normalized Cuts and Image Segmentation. *IEEE Transactions on Pattern Analysis and Machine Intelligence (PAMI)*, 22(8):888–905, 2000.

[11] Daniel A. Spielman and Shang-Hua Teng. Nearly-linear time algorithms for graph partitioning, graph sparsification, and solving linear systems. In *Proceedings of the thirty-sixth annual ACM symposium on Theory of computing*, pages 81–90, 2004.

[12] Daniel A. Spielman and Shang-Hua Teng. A local clustering algorithm for massive graphs and its application to nearly linear time graph partitioning. *SIAM Journal on Computing*, 42(1):1–26, 2013.

[13] U. von Luxburg. A tutorial on spectral clustering. *Statistics and Computing*, 17(4):395–416, 2007.

[14] Zhirong Yang, Tele Hao, Onur Dikmen, Xi Chen, and Erkki Oja. Clustering by nonnegative matrix factorization using graph random walk. In *Advances in Neural Information Processing Systems (NIPS)*, pages 1088–1096, 2012.

[15] Stella X. Yu and Jianbo Shi. Multiclass spectral clustering. in international conference on computer vision. In *International Conference on Computer Vision*, 2003.

